# Predicting the Optimal Spacing of Study:
# A Multiscale Context Model of Memory

**Michael C. Mozer[*], Harold Pashler[†], Nicholas Cepeda[°],**
**Robert Lindsey[*], & Ed Vul[‡]**
[*] Dept. of Computer Science, University of Colorado
[†]Dept. of Psychology, UCSD
[°]Dept. of Psychology, York University
[‡]Dept. of Brain and Cognitive Sciences, MIT

## Abstract

When individuals learn facts (e.g., foreign language vocabulary) over multiple study sessions, the temporal spacing of study has a significant impact on memory retention. Behavioral experiments have shown a nonmonotonic relationship between spacing and retention: short or long intervals between study sessions yield lower cued-recall accuracy than intermediate intervals. Appropriate spacing of study can double retention on educationally relevant time scales. We introduce a Multiscale Context Model (MCM) that is able to predict the influence of a particular study schedule on retention for specific material. MCM's prediction is based on empirical data characterizing forgetting of the material following a single study session. MCM is a synthesis of two existing memory models (Staddon, Chelaru, & Higa, 2002; Raaijmakers, 2003). On the surface, these models are unrelated and incompatible, but we show they share a core feature that allows them to be integrated. MCM can determine study schedules that maximize the durability of learning, and has implications for education and training. MCM can be cast either as a neural network with inputs that fluctuate over time, or as a cascade of leaky integrators. MCM is intriguingly similar to a Bayesian multiscale model of memory (Kording, Tenenbaum, & Shadmehr, 2007), yet MCM is better able to account for human declarative memory.

## 1   Introduction

Students often face the task of memorizing facts such as foreign language vocabulary or state capitals. To retain such information for a long time, students are advised not to cram their study, but rather to study over multiple, well-spaced sessions. This advice is based on a memory phenomenon known as the *distributed practice* or *spacing* effect (Cepeda, Pashler, Vul, Wixted, & Rohrer, 2006).

The spacing effect is typically studied via a controlled experimental paradigm in which participants are asked to study unfamiliar paired associates (e.g., English-Japanese vocabulary) in two sessions. The time between sessions, known as the *intersession interval* or *ISI*, is manipulated across participants. Some time after the second study session, a cued-recall test is administered to the participants, e.g., "What is 'rabbit' in Japanese?" The lag between second session and the test is known as the *retention interval* or *RI*.

Recall accuracy as a function of ISI follows a characteristic curve. The solid line of Figure 1a sketches this curve, which we will refer to as the *spacing function*. The left edge of the graph corresponds to massed practice, when session two immediately follows session one. Recall accuracy rises dramatically as the ISI increases, reaches a peak, and falls off gradually. The ISI corresponding to the peak—the *optimal ISI*—depends strongly on RI: a meta-analysis by Cepeda et al. (2006) sug-

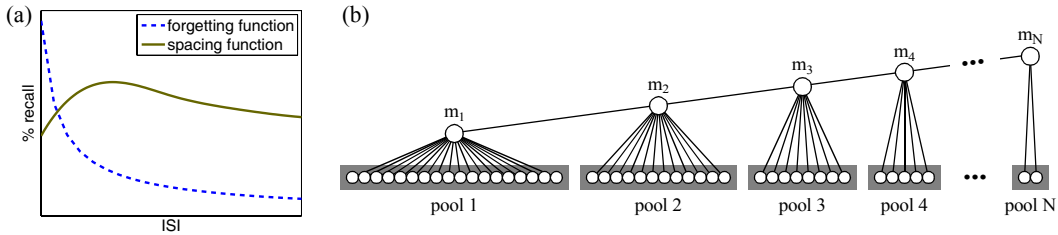

Figure 1: (a) The spacing function (solid line) depicts recall at test following two study sessions separated by a given ISI; the forgetting function (dashed line) depicts recall as a function of the lag between study and test. (b) A sketch of the Multiscale Context Model.

gests a power-law relationship. The optimal ISI almost certainly depends on the specific materials being studied and the manner of study as well. For educationally relevant RIs on the order of weeks and months, the effect of spacing can be tremendous: optimal spacing can double retention over massed practice (Cepeda et al., in press).

The spacing function is related to another observable measure of retention, the *forgetting function*, which characterizes recall accuracy following a single study session as a function of the lag between study and test. For example, suppose participants in the experiment described above learned material in study session 1, and were then tested on the material immediately prior to study session 2. As the ISI increased, session 1 memories would decay. This decay is shown in the dashed line of Figure 1a. Typical forgetting functions follow a generalized power-law decay, of the form $P(\text{recall}) = A(1 + Bt)^{-C}$, where $A$, $B$, and $C$ are constants, and $t$ is the study-test lag (Wixted & Carpenter, 2007).

Our goal is to develop a model of long-term memory that characterizes the memory-trace strength of items learned over two or more sessions. The model predicts recall accuracy as a function of the RI, taking into account the study *schedule*—the ISI or set of ISIs determining the spacing of study sessions. We would like to use this model to prescribe the optimal study schedule.

The spacing effect is among the best known phenomena in cognitive psychology, and many theoretical explanations have been suggested. Two well developed computational models of human memory have been elaborated to explain the spacing effect (Pavlik & Anderson, 2005; Raaijmakers, 2003). These models are necessarily complex: the brain contains multiple, interacting memory systems whose decay and interference characteristics depend on the specific content being stored and its relationship to other content. Consequently, these computational theories are fairly flexible and can provide reasonable post-hoc fits to spacing effect data, but we question their predictive value.

Rather than developing a general theory of memory, we introduce a model that specifically predicts the shape of the spacing function. Because the spacing function depends not only on the RI, but also on the nature of the material being learned, and the manner and amount of study, the model requires empirical constraints. We propose a novel approach to obtaining a predictive model: we collect behavioral data to determine the forgetting function for the specific material being learned. We then use the forgetting function, which is based on a single study session, to predict the spacing function, which is based on two or more study sessions. Such a predictive model has significant implications for education and training. The model can be used to search for the ISI or set of ISIs that maximizes expected recall accuracy for a fixed RI. Although the required RI is not known in practical settings, one can instead optimize over RI as a random variable with an assumed distribution.

## 2 Accounts of the spacing effect

We review two existing theories proposed to explain the spacing effect, and then propose a synthesis of these theories. The two theories appear to be unrelated and mutually exclusive on the surface, but in fact share a core unifying feature. In contrast to most modeling work appearing in the NIPS volumes, our model is cast at Marr's implementation level, not at the level of a computational theory. However, after introducing our model and showing its predictive power, we discuss an intriguingly similar Bayesian theory of memory adaptation (Kording et al., 2007). Although our model has a

strong correspondence with the Bayesian model, their points of difference seem to be crucial for predicting behavioral phenomena of human declarative memory.

## 2.1 Encoding-variability theories

One class of theories proposed to explain the spacing effect focuses on the notion of *encoding variability*. According to these theories, when an item is studied, a memory trace is formed that incorporates the current *psychological context*. Psychological context includes conditions of study, internal state of the learner, and recent experiences of the learner. Retrieval of a stored item depends at least in part on the similarity of the contexts at the study and test. If psychological context is assumed to fluctuate randomly over time, two study sessions close together in time will have similar contexts. Consequently, at the time of a recall test, either both study contexts will match the test context or neither will. Increasing the ISI can thus prove advantageous because the test context will have higher likelihood of matching one study context or the other. Greater contextual variation enhances memory on this account by making for less redundancy in the underlying memory traces. However, increasing the ISI also incurs a retrieval cost because random drift makes the first-study context increasingly less likely to match the test context. The optimal ISI depends on the tradeoff between the retrieval benefit and cost at test.

Raaijmakers (2003) developed an encoding variability theory by incorporating time-varying contextual drift into the well-known Search of Associative Memory (SAM) model (Raaijmakers & Shiffrin, 1981), and explained a range of data from the spacing literature. In this model, the contextual state is characterized by a high-dimensional binary vector. Each element of the vector indicates the presence or absence of a particular contextual feature. The contextual state evolves according to a stochastic process in which features flip from absent to present at rate $\pi_{01}$ and from present to absent at rate $\pi_{10}$. If the context is sampled at two points in time with lag $\Delta t$, the probability that a contextual feature will be present at both times is

$$P(\text{feature present at time } t \text{ and } t + \Delta t) = \beta^2 + \beta(1 - \beta)\exp(-\Delta t/\tau), \tag{1}$$

where $\tau \equiv 1/(\pi_{01} + \pi_{10})$ and $\beta \equiv \pi_{01}\tau$ is the expected proportion of features present at any instant.

To assist in understanding the mechanisms of SAM, we find it useful to recast the model as a neural network. The input layer to this neural net is a pool of binary valued neurons that represent the contextual state at the current time; the output layer consists of a set of *memory* elements, one per item to be stored. To simplify notation throughout this paper, we'll describe this model and all others in terms of a single-item memory, allowing us to avoid an explicit index term for the item being stored or retrieved. The memory element for the item under consideration has an activation level, $m$, which is a linear function of the context unit activities: $m = \sum_j w_j c_j$, where $c_j$ is the binary activation level of context unit $j$ and $w_j$ is the strength of connection from context $j$. The probability of retrieval of the item is assumed to be monotonically related to $m$.

When an item is studied, its connection strengths are adjusted according to a Hebbian learning rule with an upper limit on the connection strength:

$$\Delta w_j = \min(1 - w_j, \ c_j \hat{m}), \tag{2}$$

where $\hat{m} = 1$ if the item was just presented for study, or 0 otherwise. When an item is studied, the weights for all contextual features present at the time of study will be strengthened. Later retrieval is more likely if the context at test matches the context at study: the memory element receives a contribution only when an input is active and its connection strength is nonzero. Thus, after a single study and lag $\Delta t$, retrieval probability is directly related to Equation 1. When an item has been studied twice, retrieval will be more robust if the two study opportunities strengthen different weights, which occurs when the ISI is large and the contextual states do not overlap significantly.

One other feature of SAM is crucial for explaining spacing-effect data. After an item has been studied at least once, SAM assumes that the memory trace resulting from further study is influenced by whether the item is accessible to retrieval at the time of study. Specifically, SAM assumes that the weights have effectively decayed to zero if recall fails. Other memory models similarly claim that memory traces are weaker if an item is inaccessible to retrieval at the time of study (e.g., Pavlik & Anderson, 2005), which we label as the *retrieval-dependent update* assumption.

We have described the key components of SAM that explain the spacing effect, but the model has additional complexity, including a short-term memory store, inter-item interference, and additional

context based on associativity and explicit cues. Even with all this machinery, SAM has a serious limitation. Spacing effects occur on many time scales (Cepeda et al., 2006). SAM can explain effects on any one time scale (e.g., hours), but the same model cannot explain spacing effects on a different time scale (e.g., months). The reason is essentially that the exponential decay in context overlap bounds the time scale at which the model operates.

## 2.2 Predictive-utility theories

We now turn to another class of theories that has been proposed to explain the spacing effect. These theories, which we will refer to as *predictive-utility theories*, are premised on the assumption that memory is limited in capacity and/or is imperfect and allows intrusions. To achieve optimal performance, memories should therefore be erased if they are not likely to be needed in the future. Anderson and Milson (1989) proposed a rational analysis of memory from which they estimated the future need probability of a stored trace. When an item is studied multiple times with a given ISI, the rational analysis suggests that the need probability drops off rapidly following the last study once an interval of time greater than the ISI has passed. Consequently, increasing the ISI should lead to a more persistent memory trace. Although this analysis yields a reasonable qualitative match to spacing-effect data, no attempt was made to make quantitative predictions.

The notion of predictive utility is embedded in the *multiple time-scale* or *MTS* model of Staddon et al. (2002). In MTS, each item to be stored is represented by a dedicated cascade of $N$ leaky integrators. The activation of integrator $i$, $x_i$, decays over time according to:

$$x_i(t + \Delta t) = x_i(t) \exp(-\Delta t / \tau_i), \tag{3}$$

where $\tau_i$ is the decay time constant. The probability of retrieving the item is related to the total trace strength, $s_N$, where $s_k = \sum_{j=1}^{k} x_j$. The integrators are ordered from shortest to longest time constant, i.e., $\tau_i < \tau_{i+1}$ for all $i$. When an item is studied, the integrators receive a bump in activity according to a *cascaded error-correction* update,

$$\Delta x_i = \epsilon \max(0, 1 - s_{i-1}), \tag{4}$$

which is based on the idea that an integrator at some time scale $\tau_i$ receives a boost only if integrators at shorter time scales fail to represent the item at the time it is studied. The constant $\epsilon$ is a step size. When an item is repeatedly presented for study with short ISIs, the trace can successfully be represented by the integrators with short time constants, and consequently, the trace will decay rapidly. Increasing the spacing shifts the representation to integrators with slower decay rates.

MTS was designed to explain rate-sensitive habituation data from the animal learning literature: the fact that recovery following spaced stimuli is slower than following massed. We tried fitting MTS to human-memory data and were unable to obtain quantitatively accurate fits.

## 3 The multiscale context model (MCM)

SAM and MTS are motivated by quite different considerations, and appear to be unrelated mechanisms. Nonetheless, they share a fundamental property: both suppose an exponential decay of internal representations over time (compare Equations 1 and 3). When we establish a correspondence between the mechanisms in SAM and MTS that produce exponential decay, we obtain a synthesis of the two models that incorporates features of each. Essentially, we take from SAM the notion of contextual drift and retrieval-dependent update, and from MTS the multiscale representation and the cascaded error-correction memory update, and we obtain a new model which we call the *Multiscale Context Model* or *MCM*. MCM can be described as a neural network whose input layer consists of $N$ pools of time-varying context units. Units in pool $i$ operate with time constant $\tau_i$. The relative size of pool $i$ is $\gamma_i$. MCM is thus like SAM with multiple pools of context units. MCM can also be described in terms of $N$ leaky integrators, where integrator $i$ has time constant $\tau_i$ and activity scaled by $\gamma_i$. MCM is thus like MTS with the addition of scaling factors.

Before formally describing MCM, we detour to explain the choice of the parameters $\{\tau_i\}$ and $\{\gamma_i\}$. As the reader might infer from our description of SAM and MTS, these parameters characterize memory decay, extending Equation 3 such that the total trace strength at time $t$ is defined as:

$$s_N(t) = \sum_{i=1}^{N} \gamma_i \exp(-\frac{t}{\tau_i}) x_i(0).$$

If $x_i(0) = 1$ for all $i$—which is the integrator activity following the first study in MTS—the trace strength as a function of time is a mixture of exponentials. To match the form of human forgetting (Figure 1), this mixture must approximate a power function. We can show that a generalized power function can be exactly expressed as an infinite mixture of exponentials:

$$A(1 + Bt)^{-C} = A \int_0^\infty \text{Inv-Gamma}(\tau; C, 1) \exp(\frac{Bt}{\tau}) d\tau,$$

where $\text{Inv-Gamma}(\tau; C, 1)$ is the inverse-gamma probability density function with shape parameter C and scale 1, and the equality is valid for $t \geq 0$ and $C > 0$. We have identified several finite mixture-of-exponential formulations that empirically yield an extremely good approximation to arbitrary power functions over ten orders of magnitude. The formulation we prefer defines $\tau_i$ and $\gamma_i$ in terms of four primitive parameters:

$$\tau_i = \mu\nu^i \quad \text{and} \quad \gamma_i = \omega\xi^i / \sum_{j=1}^N \xi^j. \tag{5}$$

With $\nu > 1$ and $\xi < 1$, the higher-order components (i.e., larger indices) represent exponentially longer time scales with exponentially smaller weighting. As a result, truncating higher-order mixture components has little impact on the approximation on shorter time scales. Consequently, we simply need to pick a value of $N$ that allows for a representation of many orders of magnitude of time. Given $N$ and human forgetting data collected in an experiment, we can search for the parameters $\{\mu, \nu, \omega, \xi\}$ that obtain a least squares fit to the data. Given the human forgetting function function, then, we can completely determine the $\{\tau_i\}$ and $\{\gamma_i\}$. In all simulation results we report, we fixed $N = 100$, although equivalent results are obtained for $N = 50$ or $N = 200$.

## 3.1 Casting MCM as a cascade of leaky integrators

Assume that—as in MTS—a dedicated set of $N$ leaky integrators hold the memory of each item to be learned. Let $x_i$ denote the activity of integrator $i$ associated with the item, and let $s_i$ be the average strength of the first $i$ integrators, weighted by the $\{\gamma_j\}$ terms:

$$s_i = \frac{1}{\Gamma_i} \sum_{j=1}^i \gamma_j x_j, \text{ where } \Gamma_i = \sum_{j=1}^i \gamma_j.$$

The recall probability is simply related to the net strength of the item: $P(\text{recall}) = \min(1, s_N)$.

When an item is studied, its integrators receive a boost in activity. Integrator $i$ receives a boost that depends on how close the average strength of the first $i$ integrators is to full strength, i.e.,

$$\Delta x_i = \epsilon(1 - s_i) \tag{6}$$

where $\epsilon$ is a step size. We adopt the retrieval-dependent update assumption of SAM, and fix $\epsilon = 1$ for an item that is unsuccessfully recalled at the time of study, and $\epsilon = \epsilon_r > 1$ for an item that is successfully recalled.

This description of MCM is identical to MTS except the following. (1) MTS weighs all integrators equally when combining the individual integrator activities. MCM uses a $\gamma$-weighted average. (2) MTS provides no guidance in setting the $\tau$ and $\gamma$ constants; MCM constrains these parameters based on the human forgetting function. (3) The integrator update magnitude is retrieval dependent, as in SAM. (4) The MCM update rule (Equation 6) is based on $s_i$, whereas the MTS rule (Equation 4) is based on $s_{i-1}$. This modification is motivated by the neural net formulation of MCM, in which using $s_i$ allows the update to be interpreted as performing gradient ascent in prediction ability.

## 3.2 Casting MCM as a neural network

The neural net conceptualization of MCM is depicted in Figure 1b. The input layer is like that of SAM with the context units arranged in $N$ pools, with $\gamma_i$ being the relative size of pool $i$. The activity of unit $j$ in pool $i$ is denoted $c_{ij}$. The context units are binary valued and units in pool $i$ flip with time constant $\tau_i$. On average a fraction $\beta$ are on at any time. ($\beta$ has no effect on the model's predictions, and is cancelled out in the formulation that follows.)

As depicted in Figure 1b, the model also includes a set of $N$ *memory elements* for each item to be learned. Memory elements are in one-to-one correspondence with context pools. Activation of memory element $i$, denoted $m_i$, indicates strength of retrieval for the item based on context pools $1...i$. The activation function is cascaded such that memory element $i$ receives input from context units in pool $i$ as well as memory element $i-1$:

$$m_i = m_{i-1} + \sum_j w_{ij}c_{ij} + b,$$

where $w_{ij}$ is the connection weight from context unit $j$ to memory element $i$, $m_0 \equiv 0$, and $b = -\beta/(1-\beta)$ is a bias weight. The bias simply serves to offset spurious activity reaching the memory elements, activity that is unrelated to the fact that the item was previously studied and stored. The larger the fraction of context units that are on at any time ($\beta$), the more spurious activation there will be that needs to be cancelled out. The probability of recalling the item is related to the activity of memory element $N$: $P(\text{recall}) = \min(1, m_N)$.

When the item is studied, the weights from context units in pool $i$ are adjusted according to an update rule that performs gradient descent in an error measure $E_i = e_i^2$, where $e_i = 1 - m_i/\Gamma_i$. This error is minimized when the memory element $i$ reaches activation level $\Gamma_i$ (defined earlier as the proportion of units in the entire context pool that contributes to activity at stage $i$). The weight update that performs gradient descent in $E_i$ is

$$\Delta w_{ij} = \frac{\epsilon}{N\beta(1-\beta)} e_i c_{ij}, \tag{7}$$

where $\epsilon$ is a learning rate and the denominator of the first term is a normalization constant which can be folded into the learning rate. As in SAM, $\epsilon$ is assumed to be contingent on retrieval success at the start of the study trial, in the manner we described previously.

What is the motivation for minimizing the prediction error at every stage, versus minimizing the prediction error just at the final stage, $E_N$? To answer this question, note that there are two consequences of minimizing the error $E_i$ to zero for any $i$. First, reducing $E_i$ will also likely serve to reduce $E_l$ for all $l > i$. Second, achieving this objective will allow the $\{w_{l,j,k} : l > i\}$ to all be set to zero without any effect on the memory. Essentially, there is no need to store information for a longer time scale than it is needed.

This description of MCM is identical to SAM except: (1) SAM has a single temporal scale of representation; MCM has a multiscale representation. (2) SAM's memory update rule can be interpreted as Hebbian learning; MCM's update can be interpreted as error-correction learning.

### 3.3   Relating leaky integrator and neural net characterizations of MCM

To make contact with MTS, we have described MCM as a cascade of leaky integrators, and to make contact with SAM, we have described MCM as a neural net. One can easily verify that the leaky-integrator and neural-net descriptions of MCM are equivalent via the following correspondence between variables of the two models, where $E[.]$ denotes the expectation over context representations:

$$s_i = E[m_i]/\Gamma_i \quad \text{and} \quad x_i = \frac{\sum_j E[w_{ij}c_{ij}] + b}{N\beta(1-\beta)}.$$

## 4   Simulations

Cepeda and colleagues (Cepeda, Vul, Rohrer, Wixted, & Pashler, 2008; Cepeda et al., in press) have recently conducted well-controlled experimental manipulations of spacing involving RIs on educationally relevant time scales of days to months. Most research in the spacing literature involves brief RIs, on the scale of minutes to an hour, and methodological concerns have been raised with the few well-known studies involving longer RIs (Cepeda et al., 2006). In Cepeda's experiments, participants study a set of paired associates over two sessions. In the first session, participants are trained until they reach a performance criterion, ensuring that the material has been successfully encoded. At the start of the second session, participants are tested via a cued-recall paradigm, and then are given a fixed number of study passes through all the pairs. Following a specified RI, a final cued-recall test is administered. Recall accuracy at the start of the second session provides the basic forgetting function, and recall accuracy at test provides the spacing function.

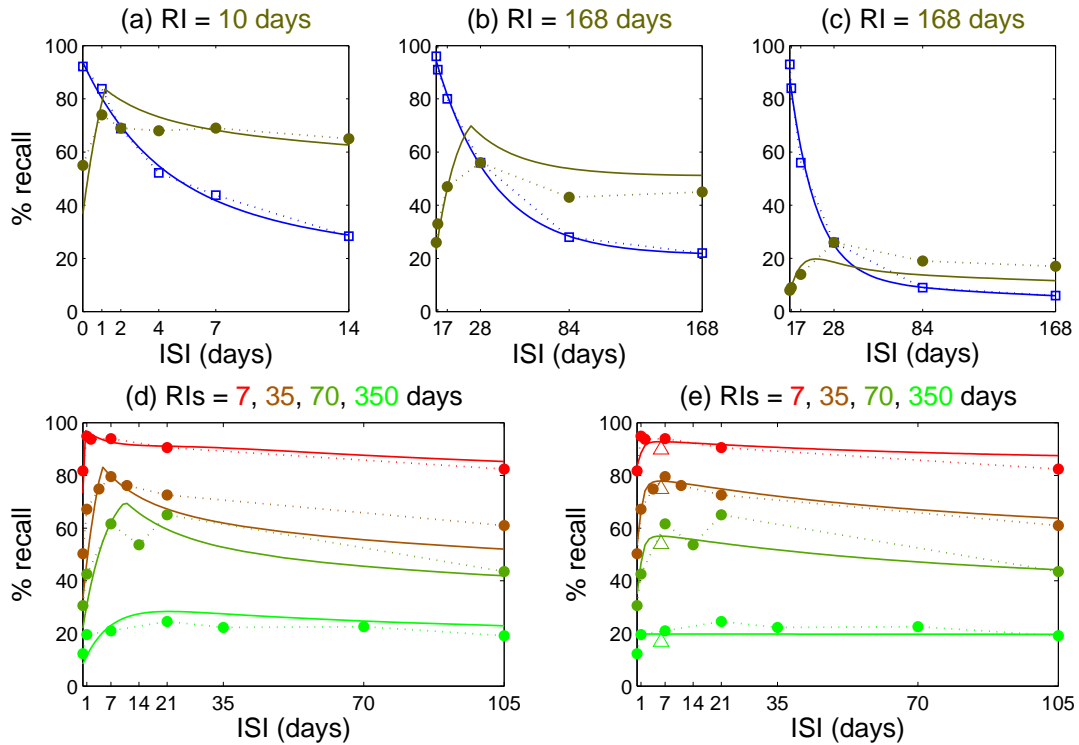

Figure 2: Modeling and experimental data of (Cepeda et al., in press) (a) Experiment 1 (Swahili-English), (b) Experiment 2a (obscure facts), and (c) Experiment 2b (object names). The four RI conditions of Cepeda et al. (2008) are modeled using (d) MCM and (e) the Bayesian multiscale model of Kording et al. (2007). In panel (e), the peaks of the model's spacing functions are indicated by the triangle pointers.

For each experiment, we optimized MCM's parameters, $\{\mu, \nu, \omega, \xi\}$, to obtain a least squares fit to the forgetting function. These four model parameters determine the time constants and weighting coefficients of the mixture-of-exponentials approximation to the forgetting function (Equation 5). The model has only one other free parameter, $\epsilon_r$, the magnitude of update on a trial when an item is successfully recalled (see Equation 6). We chose $\epsilon_r = 9$ for all experiments, based on hand tuning the parameter to fit the first experiment reported here. With $\epsilon_r$, MCM is fully constrained and can make strong predictions regarding the spacing function.

Figure 2 shows MCM's predictions of Cepeda's experiments. Panels a-c show the forgetting function data for the experiments (open blue squares connected by dotted lines), MCM's post-hoc fit to the forgetting function (solid blue line), the spacing function data (solid green points connected by dotted lines), and MCM's parameter-free prediction of the spacing function (solid green line). The individual panels show the ISIs studied and the RI. For each experiment, MCM's prediction of the peak of the spacing function is entirely consistent with the data, and for the most part, MCM's quantiative predictions are excellent. (In panel c, MCM's predictions are about 20% too low across the range of ISIs.) Interestingly, the experiments in panels b and c explored identical ISIs and RIs with two different types of material. With the coarse range of ISIs explored, the authors of these experiments concluded that the peak ISI was the same independent of the material (28 days). MCM suggests a different peak for the two sets of material, a prediction that can be evaluated empirically. (It would be extremely surprising to psychologists if the peak were in general independent of the material, as content effects pervade the memory literature.)

Panel d presents the results of a complex study involving a single set of items studied with 11 different ISIs, ranging from minutes to months, and four RIs, ranging from a week to nearly a year. We omit the fit to the forgetting function to avoid cluttering the graph. The data and model predictions

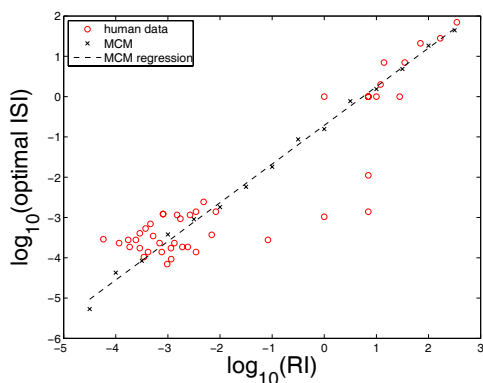

Figure 3: A meta-analysis of the literature by Cepeda et al. (2006). Each red circle represents a single spacing experiment in which the ISI was varied for a given RI. The optimal ISI obtained in the experiment is plotted against the RI on a log-log scale. (Note that the data are intrinsically noisy because experiments typically examine only a small set of ISIs, from which the 'optimum' is chosen.) The X's represent the mean from 1000 replications of MCM for a given RI with randomly drawn parameter settings (i.e., random forgetting functions), and the dashed line is the best regression fit to the X's. Both the experimental data and MCM show a power law relationship between optimal ISI and RI.

are color coded by RI, with higher recall accuracy for shorter RIs. MCM predicts the spacing functions with absolutely spectacular precision, considering the predictions are fully constrained and parameter free. Moreover, MCM anticipates the peaks of the spacing functions, with the curvature of the peak decreasing with the RI, and the optimal ISI increasing with the RI.

In addition to these results, MCM also predicts the probability of recall at test conditional on successful or unsuccessful recall during the test at the start of the second study session. As explained in Figure 3, MCM obtains a sensible parameter-free fit to a meta-analysis of the experimental literature by Cepeda et al. (2006). Finally, MCM is able to post-hoc fit classic studies from the spacing literature (for which forgetting functions are not available).

# 5   Discussion

MCM's blind prediction of 7 different spacing functions is remarkable considering that the domain's complexity (the content, manner and amount of study) is reduced to four parameters, which are fully determined by the forgetting function. Obtaining empirical forgetting functions is straightforward. Obtaining empirical evidence to optimize study schedules, especially when more than two sessions are involved, is nearly infeasible. MCM thus offers a significant practical tool for educators in devising study schedules. Optimizing study schedules with MCM is straightfoward, and particularly useful considering that MCM can optimize not only for a known RI but for RI as a random variable.

MCM arose from two existing models, MTS and SAM, and all three models are characterized at Marr's implementation or algorithmic levels, not at the level of a computational theory. Kording et al. (2007) have proposed a Bayesian memory model which has intriguing similarities to MCM, and has the potential of serving as the complementary computational theory. The model is a Kalman filter (KF) with internal state variables that decay exponentially at different rates. The state predicts the appearance of an item in the temporal stream of experience. The dynamics of MCM can be exactly mapped onto the KF, with $\tau$ related to the decay of a variable, and $\gamma$ to its internal noise level. However, the KF model has a very different update rule, based on the Kalman gain. We have tried to fit experimental data with the KF model, but have not been satisfied with the outcome. For example, Figure 2e shows a least-squares fit to the six free parameters of the KF model to the Cepeda et al. (2008) data. (Two parameters determine the range of time scales; two specify internal and observation noise levels; and two perform an affine transform from internal memory strength to recall probability.) In terms of sum-squared error, the model shows a reasonable fit, but the model clearly misses the peaks of the spacing functions, and in fact predicts a peak that is independent of RI. Notably, the KF model is a post-hoc fit to the spacing functions, whereas MCM produces a true prediction of the spacing functions, i.e., parameters of MCM are determined without peeking at the spacing function. Exploring many parameterizations of the KF model, we find that the model generally predicts decreasing or constant optimal ISIs as a function of the RI. In contrast, MCM necessarily produces an increasing optimal ISI as a function of the RI, consistent with all behavioral data. It remains an important and intriguing challenge to unify MCM and the KF model; each has something to offer the other.

**References**

Anderson, J. R., & Milson, R. (1989). Human memory: An adaptive perspective. *Psych. Rev.*, *96*, 703–719.

Cepeda, N. J., Coburn, N., Rohrer, D., Wixted, J. T., Mozer, M. C., & Pashler, H. (in press). Optimizing distributed practice: Theoretical analysis and practical implications. *Journal of Experimental Psychology*.

Cepeda, N. J., Pashler, H., Vul, E., Wixted, J. T., & Rohrer, D. (2006). Distributed practice in verbal recall tasks: A review and quantitative synthesis. *Psychological Bulletin*, *132*, 354–380.

Cepeda, N. J., Vul, E., Rohrer, D., Wixted, J. T., & Pashler, H. (2008). Spacing effects in learning: A temporal ridgeline of optimal retention. *Psychological Science*, *19*, 1095–1102.

Kording, K. P., Tenenbaum, J. B., & Shadmehr, R. (2007). The dynamics of memory as a consequence of optimal adaptation to a changing body. *Nature Neuroscience*, *10*, 779–786.

Pavlik, P. I., & Anderson, J. R. (2005). Practice and forgetting effects on vocabulary memory: An activation-based model of the spacing effect. *Cognitive Science*, *29*(4), 559-586.

Raaijmakers, J. G. W. (2003). Spacing and repetition effects in human memory: application of the SAM model. *Cognitive Science*, *27*, 431–452.

Raaijmakers, J. G. W., & Shiffrin, R. M. (1981). Search of associative memory. *Psych. Rev.*, *88*, 93–134.

Staddon, J. E. R., Chelaru, I. M., & Higa, J. J. (2002). Habituation, memory and the brain: The dynamics of interval timing. *Behavioural Processes*, *57*, 71-88.

Wixted, J. T., & Carpenter, S. K. (2007). The Wickelgren power law and the Ebbinghaus savings function. *Psychological Science*, *18*, 133–134.

